# Learning Informative Statistics: A Nonparametric Approach

**John W. Fisher III, Alexander T. Ihler, and Paul A. Viola**
Massachusetts Institute of Technology
77 Massachusetts Ave., 35-421
Cambridge, MA 02139
{*fisher,ihler,viola*}*@ai.mit.edu*

## Abstract

We discuss an information theoretic approach for categorizing and modeling dynamic processes. The approach can learn a compact and informative statistic which summarizes past states to predict future observations. Furthermore, the uncertainty of the prediction is characterized nonparametrically by a joint density over the learned statistic and present observation. We discuss the application of the technique to both noise driven dynamical systems and random processes sampled from a density which is conditioned on the past. In the first case we show results in which both the dynamics of random walk and the statistics of the driving noise are captured. In the second case we present results in which a summarizing statistic is learned on noisy random telegraph waves with differing dependencies on past states. In both cases the algorithm yields a principled approach for discriminating processes with differing dynamics and/or dependencies. The method is grounded in ideas from information theory and nonparametric statistics.

## 1 Introduction

Noisy dynamical processes abound in the world – human speech, the frequency of sun spots, and the stock market are common examples. These processes can be difficult to model and categorize because current observations are dependent on the past in complex ways. Classical models come in two sorts: those that assume that the dynamics are linear and the noise is Gaussian (e.g. Weiner etc.); and those that assume that the dynamics are discrete (e.g. HMM's). These approach are wildly popular because they are tractable and well understood. Unfortunately there are many processes where the underlying theoretical assumptions of these models are false. For example we may wish to analyze a system with linear dynamics and non-Gaussian noise or we may wish to model a system with an unknown number of discrete states.

We present an information-theoretic approach for analyzing stochastic dynamic processes which can model simple processes like those mentioned above, while retaining the flexibility to model a wider range of more complex processes. The key insight is that we can often learn a simplifying informative statistic of the past from samples using nonparametric estimates of both entropy and mutual information. Within this framework we can predict future states and, of equal importance, characterize the uncertainty accompanying those

predictions. This non-parametric model is flexible enough to describe uncertainty which is more complex than second-order statistics. In contrast techniques which use squared prediction error to drive learning are focused on the mode of the distribution.

Taking an example from financial forecasting, while the *most likely* sequence of pricing events is of interest, one would also like to know the accompanying distribution of price values (i.e. even if the most likely outcome is appreciation in the price of an asset, knowledge of lower, but not insignificant, probability of depreciation is also valuable). Towards that end we describe an approach that allows us to simultaneously learn the dependencies of the process on the past as well as the uncertainty of future states. Our approach is novel in that we fold in concepts from information theory, nonparametric statistics, and learning.

In the two types of stochastic processes we will consider, the challenge is to summarize the past in an efficient way. In the absence of a known dynamical or probabilistic model, can we learn an informative statistic (ideally a sufficient statistic) of the past which minimizes our uncertainty about future states? In the classical linear state-space approach, uncertainty is characterized by mean squared error (MSE) which implicitly assume Gaussian statistics. There are, however, linear systems with interesting behavior due to non-Gaussian statistics which violate the assumption underlying MSE. There are also nonlinear systems and purely probabilistic processes which exhibit complex behavior and are poorly characterized by mean square error and/or the assumption of Gaussian noise.

Our approach is applicable to both types of processes. Because it is based on nonparametric statistics we characterize the uncertainty of predictions in a very general way: by a density of possible future states. Consequently the resulting system captures both the dynamics of the systems (through a parameterization) and the statistics of driving noise (through a nonparametric modeling). The model can then be used to classify new signals and make predictions about the future.

## 2   Learning from Stationary Processes

In this paper we will consider two related types of stochastic processes, depicted in figure 1. These processes differ in how current observations are related to the past. The first type of process, described by the following set of equations, is a discrete time dynamical (possibly nonlinear) system:

$$x_k = G(\{x_{k-1}\}_N; w_g) + \eta_k \qquad ; \{x_k\}_N = \{x_k, \ldots, x_{k-(N-1)}\} \qquad (1)$$

where, $x_k$, the state of the process at time $k$, is a function of the $N$ previous states and the present value of $\eta$. In general the sequence $\{x_k\}$ is not stationary (in the strict sense); however, under fairly mild conditions on $\{\eta_k\}$, namely that $\{\eta_k\}$ is a sequence of i.i.d. random variables (which we will always assume to be true), the sequence:

$$\epsilon_k = x_k - G(\{x_{k-1}\}_N; w_g) \qquad (2)$$

*is* stationary. Often termed an innovation sequence, for our purpose the stationarity of 2 will suffice. This leads to a prediction framework for estimating the dynamical parameters, $w_g$, of the system and to which we will adjoin a nonparametric characterization of uncertainty.

The second type of process we consider is described by a conditional probability density:

$$x_k \sim p(x_k \| \{x_{k-1}\}_N) \qquad (3)$$

In this case it is only the conditional statistics of $\{x_k\}$ that we are concerned with and they are, by definition, constant.

## 3   Learning Informative Statistics with Nonparametric Estimators

We propose to determine the system parameters by minimizing the entropy of the error residuals for systems of type (a). Parametric entropy optimization approaches have been

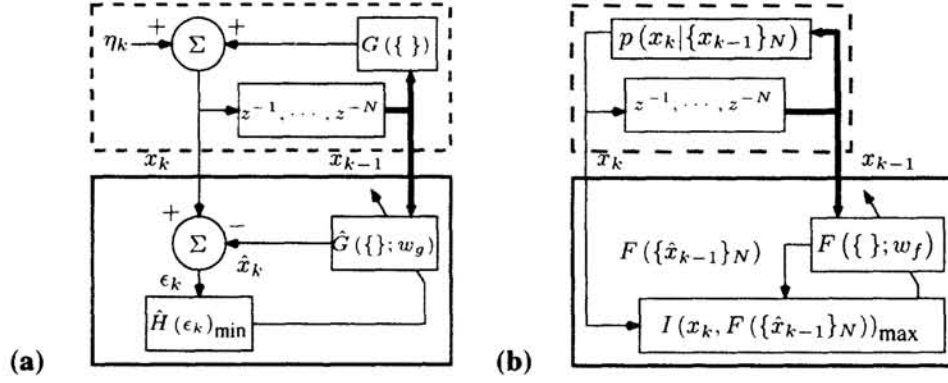

Figure 1: Two related systems: (a) dynamical system driven by stationary noise and (b) probabilistic system dependent on the finite past. Dotted box indicates source of stochastic process, while solid box indicates learning algorithm

proposed (e.g. [4]), the novelty of our approach; however, is that we estimate entropy nonparametrically. That is,

$$\hat{w}_g = \arg\min_{w_g} \hat{H}(\epsilon) \quad \hat{H}(\epsilon) \approx \int \hat{p}(\epsilon) \log \hat{p}(\epsilon) d\epsilon \quad \hat{p}(\epsilon) = -\frac{1}{M} \sum_{k=0}^{M-1} \kappa(\epsilon_k - \epsilon) \ , \ (4)$$

where the differential entropy integral is approximated using a function of the Parzen kernel density estimator [5] (in all experiments we use the Gaussian kernel). It can be shown that minimizing the entropy of the error residuals is equivalent to maximizing their likelihood [1]. In this light, the proposed criterion is seeking the maximum likelihood estimate of the system parameters using a nonparametric description of the noise density. Consequently, we solve for the system parameters and the noise density jointly.

While there is no explicit dynamical system in the second system type we do assume that the conditional statistics of the observed sequence are constant (or at worst slowly changing for an on-line learning algorithm). In this case we desire to minimize the uncertainty of predictions from future samples by summarizing information from the past. The challenge is to do so efficiently via a function of recent samples. Ideally we would like to find a sufficient statistic of the past; however, without an explicit description of the density we opt instead for an *informative* statistic. By *informative* statistic we simply mean one which reduces the conditional entropy of future samples. If the statistic were *sufficient* then the mutual information has reached a maximum [1]. As in the previous case, we propose to find such a statistic by maximizing the *nonparametric* mutual information as defined by

$$w_f = \arg\min_{w_f} \hat{I}(x_k, F(\{x_{k-1}\}_N; w_f)) \tag{5}$$

$$= \arg\min_{w_f} \hat{H}(x_k) + \hat{H}(F(\{\ \};w_f)) - \hat{H}(x_k, F(\{\ \};w_f))) \tag{6}$$

$$= \arg\min_{w_f} \hat{H}(x_k) - \hat{H}(x_k|F(\{\ \};w_f))) \tag{7}$$

By equation 6 this is equivalent to optimizing the joint and marginal entropies (which we do in practice) or, by equation 7, minimizing the conditional entropy.

We have previously presented two related methods for incorporating kernel based density estimators into an information theoretic learning framework [2, 3]. We chose the method of [3] because it provides an exact gradient of an *approximation* to entropy, but more importantly can be converted into an implicit error function thereby reducing computation cost.

## 4   Distinguishing Random Walks: An Example

In random walk the feedback function $G(\{x_{k-1}\}_1) = x_{k-1}$. The noise is assumed to be independent and identically distributed (i.i.d.). Although the sequence,$x_k$, is non-stationary the increments $(x_k - x_{k-1})$ are stationary. In this context, estimating the statistics of the residuals allows for discrimination between two random walk process with differing noise densities. Furthermore, as we will demonstrate empirically, even when one of the processes is driven by Gaussian noise (an implicit assumption of the MMSE criterion), such knowledge may not be sufficient to distinguish one process from another.

Figure 2 shows two random walk realizations and their associate noise densities (solid lines). One is driven by Gaussian noise ($\eta_k \sim N(0,1)$), while the other is driven by a bi-modal mixture of gaussians ($\eta_k \sim \frac{1}{2}N(0.95, 0.3) + \frac{1}{2}N(-0.95, 0.3)$) (note: both densities are zero-mean and unit variance). During learning, the process was modeled as fifth-order auto-regressive ($AR_5$). One hundred samples were drawn from a realization of each type and the AR parameters were estimated using the standard MMSE approach and the approach described above. With regards to parameter estimation, both methods (as expected) yield essentially the same parameters with the first coefficient being near unity and the remaining coefficients being near zero.

We are interested in the ability to distinguish one process from another. As mentioned, the current approach jointly estimates the parameters of the system as well as the density of the noise. The nonparametric estimates are shown in figure 2 (dotted lines). These estimates are then be used to compute the accumulated average log-likelihood ($L(\epsilon_k) = \frac{1}{k}\sum_{i=1}^{k} \log p(x_i)$) of the residual sequence ($\epsilon_k \approx \eta_k$) under the known and learned densities (figure 3). It is striking (but not surprising) that $L(\epsilon_k)$ of the bi-modal mixture under the Gaussian model (dashed lines, top) does not differ significantly from the Gaussian driven increments process (solid lines, top). The explanation follows from the fact that

$$\lim_{k \to \infty} L(\epsilon_k) = -(H(p_\epsilon(\epsilon)) + D(p(\epsilon)\|p_\epsilon(\epsilon))) \tag{8}$$

where $p_\epsilon(\epsilon)$ is the true density of $\epsilon$ (bi-modal), $p(\epsilon)$ is the assumed density of the likelihood test (unit-variance Gaussian), and $D(\|)$ is the Kullback-Leibler divergence [1]. In this case, $D(p(\epsilon)\|p_\epsilon(\epsilon))$ is relatively small (not true for $D(p_\epsilon(\epsilon)\|p(\epsilon))$ and $H(p_\epsilon(\epsilon))$ is less than the entropy of the unit-variance Gaussian (for fixed variance, the Gaussian density has maximum entropy). The consequence is that the likelihood test under the Gaussian assumption does not reliably distinguish the two processes. The likelihood test under the bi-modal density or its nonparametric estimate (figure 3, bottom) does distinguish the two.

The method described is not limited to linear dynamic models. It can certainly be used for nonlinear models, so long as the dynamic can be well approximated by differentiable functions. Examples for multi-layer perceptrons are described in [3].

## 5   Learning the Structure of a Noisy Random Telegraph Wave

A noisy random telegraph wave (RTW) can be described by figure 1(b). Our goal is not to demonstrate that we can analyze random telegraph waves, rather that we can robustly learn an informative statistic of the past for such a process. We define a noisy random telegraph wave as a sequence $x_k \sim N(\mu_k, \sigma)$ where $\mu_k$ is binomially distributed:

$$\mu_k \in \{\pm\mu\} \quad P\{\mu_k = -\mu_{k-1}\} = \alpha \frac{\left|\frac{1}{N}\sum_{i=1}^{N} x_{k-i}\right|}{\frac{1}{N}\sum_{i=1}^{N} |x_{k-i}|}, \tag{9}$$

$N(\mu_k, \sigma)$ is Gaussian and $\alpha < 1$. This process is interesting because the parameters are random functions of a nonlinear combination of the set $\{x_k\}_N$. Depending on the value of $N$, we observe different switching dynamics. Figure 4 shows examples of such signals for

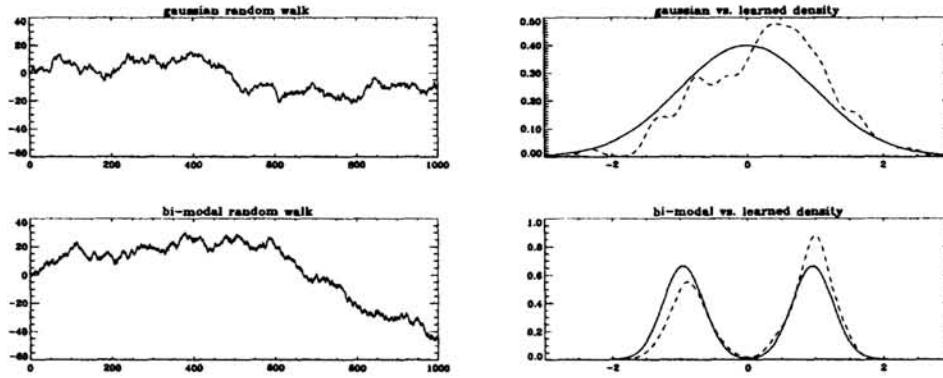

Figure 2: Random walk examples (left), comparison of known to learned densities (right).

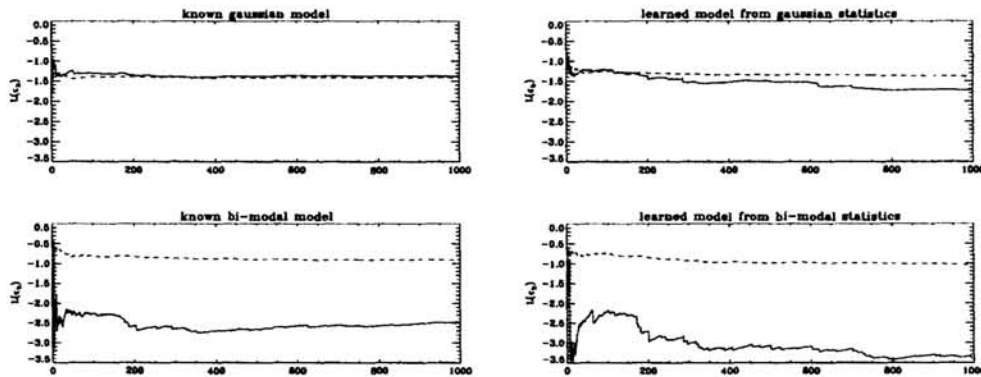

Figure 3: $L(\epsilon_k)$ under known models (left) as compared to learned models (right).

$N = 20$ (left) and $N = 4$ (right). Rapid switching dynamics are possible for both signals while $N = 20$ has periods with longer duration than $N = 4$.

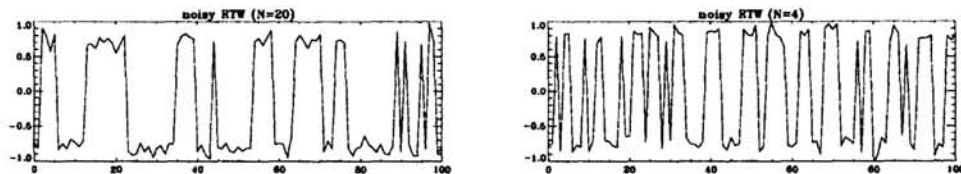

Figure 4: Noisy random telegraph wave: $N = 20$ (left), $N = 4$ (right)

In our experiments we learn a sufficient statistic which has the form

$$F(\{x_k\}_{\text{past}}) = \sigma \left( \sum_{i=1}^{M} w_{f_i} x_{k-i} \right), \tag{10}$$

where $\sigma(\ )$ is the hyperbolic tangent function (i.e. $F\{\ \}$ is a one layer perceptron). Note that a multi-layer perceptron could also be used [3].

In our experiments we train on 100 samples of noisy $\text{RTW}_{(N=20)}$ and $\text{RTW}_{(N=4)}$. We then learn statistics for each type of process using $M = \{4, 5, 15, 20, 25\}$. This tests for situations in which the depth is both under-specified and over-specified (as well as perfectly

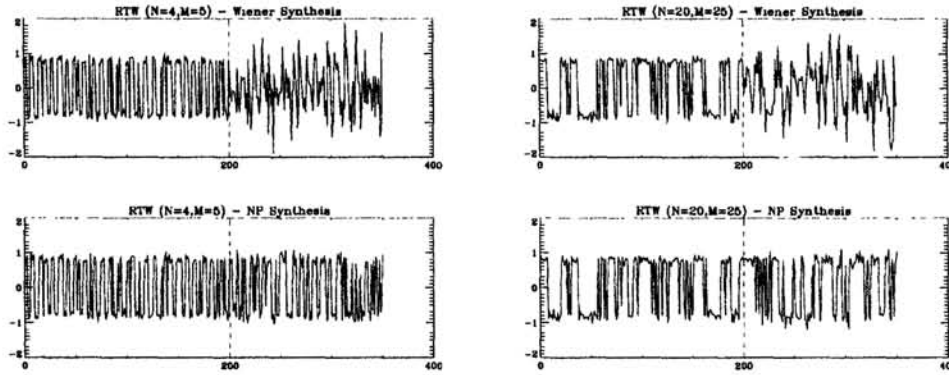

Figure 5: Comparison of Wiener filter (top) nonparametric approach (bottom) for synthesis.

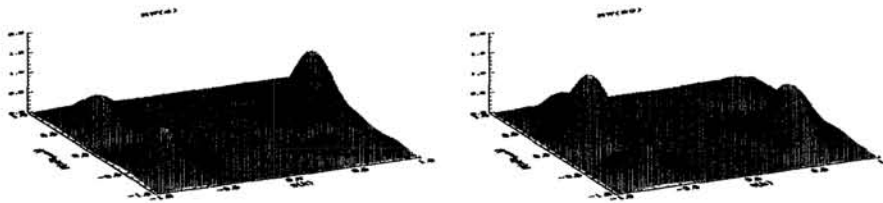

Figure 6: Informative Statistics for noisy random telegraph waves. $M = 25$ trained on $N$ equal 4 (left) and 20 (right).

specified). We will denote $F_N(\{x_k\}_M)$ as the statistic which was trained on an $\text{RTW}_{(N)}$ process with a memory depth of $M$.

Since we implicitly learn a joint density over $(x_k, F_N(\{x_k\}_M))$ synthesis is possible by sampling from that density. Figure 5 compares synthesis using the described method (bottom) to a Wiener filter (top) estimated over the same data. The results using the information theoretic approach (bottom) preserve the structure of the RTW while the Wiener filter results do not. This was achieved by collapsing the information of past samples into a single statistic (avoiding high dimension density estimation). Figure 6 shows the joint density over $(x_k, F_N(\{x_k\}_M))$ for $N = \{4, 20\}$ and $M = 25$. We see that the estimated densities are not separable and by virtue of this fact the learned statistic conveys information about the future. Figure 7 shows results from 100 monte carlo trials. In this case the depth of the statistic is matched to the process. Each plot shows the accumulated conditional log likelihood $(L(\epsilon_k) = \frac{1}{k}\sum_{i=1}^{k} \log \hat{p}(x_i|F_N(\{x_{k-1}\}_M))$ under the learned statistic with error bars. Figure 8 shows similar results after varying the memory depth $M = \{4, 5, 15, 20, 25\}$ of the statistic. The figures illustrate robustness to choice of memory depth $M$. This is not to say that memory depth doesn't matter; that is, there must be some information to exploit, but the empirical results indicate that useful information was extracted.

# 6   Conclusions

We have described a nonparametric approach for finding *informative* statistics. The approach is novel in that learning is derived from nonparametric estimators of entropy and mutual information. This allows for a means by which to 1) efficiently summarize the past, 2) predict the future and 3) characterize the uncertainty of those predictions beyond second-order statistics. Futhermore, this was accomplished without the strong assumptions accompanying parametric approaches.

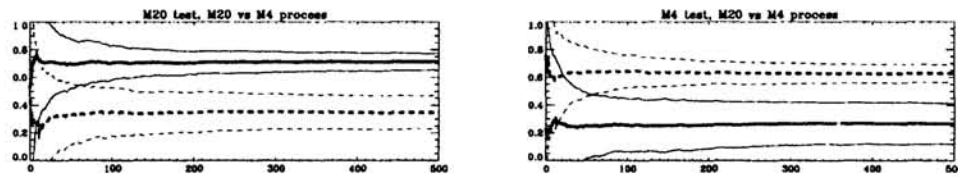

Figure 7: Conditional $L(\epsilon_k)$. Solid line indicates $\mathrm{RTW}_{(N=20)}$ while dashed line indicates $\mathrm{RTW}_{(N=4)}$. Thick lines indicate the average over all monte carlo runs while the thin lines indicate $\pm 1$ standard deviation. The left plot uses a statistic trained on $\mathrm{RTW}_{(N=20)}$ while the right plot uses a statistic trained on $\mathrm{RTW}_{(N=4)}$.

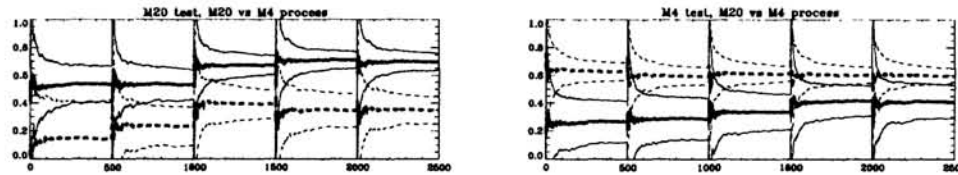

Figure 8: Repeat of figure 7 for cases with $M = \{4, 5, 15, 20, 25\}$. Obvious breaks indicate a new set of trials

We also presented empirical results which illustrated the utility of our approach. The example of random walk served as a simple illustration in learning a dynamic system in spite of the over-specification of the AR model. More importantly, we demonstrated the ability to learn both the dynamic and the statistics of the underlying noise process. This information was later used to distinguish realizations by their nonparametric densities, something not possible using MMSE error prediction.

An even more compelling result were the experiments with noisy random telegraph waves. We demonstrated the algorithms ability to learn a compact statistic which efficiently summarized the past for process identification. The method exhibited robustness to the number of parameters of the learned statistic. For example, despite overspecifying the dependence of the memory-4 in three of the cases, a useful statistic was still found. Conversely, despite the memory-20 statistic being underspecified in three of the experiments, useful information from the available past was extracted.

It is our opinion that this method provides an alternative to some of the traditional and connectionist approaches to time-series analysis. The use of nonparametric estimators adds flexibility to the class of densities which can be modeled and places less of a constraint on the exact form of the summarizing statistic.

## References

[1] T. Cover and J. Thomas. *Elements of Information Theory*. John Wiley & Sons, New York, 1991.

[2] P. Viola et al. Empricial entropy manipulation for real world problems. In Mozer Touretsky and Hasselmo, editors, *Advances in Neural Information Processing Systems*, pages ?–?, 1996.

[3] J.W. Fisher and J.C. Principe. A methodology for information theoretic feature extraction. In A. Stuberud, editor, *Proc. of the IEEE Int Joint Conf on Neural Networks*, pages ?–?, 1998.

[4] J. Kapur and H. Kesavan. *Entropy Optimization Principles with Applications*. Academic Press, New York, 1992.

[5] E. Parzen. On estimation of a probability density function and mode. *Ann. of Math Stats.*, 33:1065–1076, 1962.